# Infinite Latent Feature Models and the Indian Buffet Process

**Thomas L. Griffiths**
Cognitive and Linguistic Sciences
Brown University, Providence RI
tom_griffiths@brown.edu

**Zoubin Ghahramani**
Gatsby Computational Neuroscience Unit
University College London, London
zoubin@gatsby.ucl.ac.uk

## Abstract

We define a probability distribution over equivalence classes of binary matrices with a finite number of rows and an unbounded number of columns. This distribution is suitable for use as a prior in probabilistic models that represent objects using a potentially infinite array of features. We identify a simple generative process that results in the same distribution over equivalence classes, which we call the Indian buffet process. We illustrate the use of this distribution as a prior in an infinite latent feature model, deriving a Markov chain Monte Carlo algorithm for inference in this model and applying the algorithm to an image dataset.

## 1 Introduction

The statistical models typically used in unsupervised learning draw upon a relatively small repertoire of representations. The simplest representation, used in mixture models, associates each object with a single latent class. This approach is appropriate when objects can be partitioned into relatively homogeneous subsets. However, the properties of many objects are better captured by representing each object using multiple latent features. For instance, we could choose to represent each object as a binary vector, with entries indicating the presence or absence of each feature [1], allow each feature to take on a continuous value, representing objects with points in a latent space [2], or define a factorial model, in which each feature takes on one of a discrete set of values [3, 4].

A critical question in all of these approaches is the dimensionality of the representation: how many classes or features are needed to express the latent structure expressed by a set of objects. Often, determining the dimensionality of the representation is treated as a model selection problem, with a particular dimensionality being chosen based upon some measure of simplicity or generalization performance. This assumes that there is a single, finite-dimensional representation that correctly characterizes the properties of the observed objects. An alternative is to assume that the true dimensionality is unbounded, and that the observed objects manifest only a finite subset of classes or features [5]. This alternative is pursued in nonparametric Bayesian models, such as Dirichlet process mixture models [6, 7, 8, 9]. In a Dirichlet process mixture model, each object is assigned to a latent class, and each class is associated with a distribution over observable properties. The prior distribution over assignments of objects to classes is defined in such a way that the number of classes used by the model is bounded only by the number of objects, making Dirichlet process mixture models "infinite" mixture models [10].

The prior distribution assumed in a Dirichlet process mixture model can be specified in

terms of a sequential process called the Chinese restaurant process (CRP) [11, 12]. In the CRP, $N$ customers enter a restaurant with infinitely many tables, each with infinite seating capacity. The $i$th customer chooses an already-occupied table $k$ with probability $\frac{m_k}{i-1+\alpha}$, where $m_k$ is the number of current occupants, and chooses a new table with probability $\frac{\alpha}{i-1+\alpha}$. Customers are *exchangeable* under this process: the probability of a particular seating arrangement depends only on the number of people at each table, and not the order in which they enter the restaurant.

If we replace customers with objects and tables with classes, the CRP specifies a distribution over partitions of objects into classes. A partition is a division of the set of $N$ objects into subsets, where each object belongs to a single subset and the ordering of the subsets does not matter. Two assignments of objects to classes that result in the same division of objects correspond to the same partition. For example, if we had three objects, the class assignments $\{c_1, c_2, c_3\} = \{1, 1, 2\}$ would correspond to the same partition as $\{2, 2, 1\}$, since all that differs between these two cases is the labels of the classes. A partition thus defines an equivalence class of assignment vectors.

The distribution over partitions implied by the CRP can be derived by taking the limit of the probability of the corresponding equivalence class of assignment vectors in a model where class assignments are generated from a multinomial distribution with a Dirichlet prior [9, 10]. In this paper, we derive an infinitely exchangeable distribution over infinite binary matrices by pursuing this strategy of taking the limit of a finite model. We also describe a stochastic process (the Indian buffet process, akin to the CRP) which generates this distribution. Finally, we demonstrate how this distribution can be used as a prior in statistical models in which each object is represented by a sparse subset of an unbounded number of features. Further discussion of the properties of this distribution, some generalizations, and additional experiments, are available in the longer version of this paper [13].

## 2 A distribution on infinite binary matrices

In a latent feature model, each object is represented by a vector of latent feature values $\mathbf{f}_i$, and the observable properties of that object $\mathbf{x}_i$ are generated from a distribution determined by its latent features. Latent feature values can be continuous, as in principal component analysis (PCA) [2], or discrete, as in cooperative vector quantization (CVQ) [3, 4]. In the remainder of this section, we will assume that feature values are continuous. Using the matrix $\mathbf{F} = \begin{bmatrix} \mathbf{f}_1^T & \mathbf{f}_2^T & \cdots & \mathbf{f}_N^T \end{bmatrix}^T$ to indicate the latent feature values for all $N$ objects, the model is specified by a prior over features, $p(\mathbf{F})$, and a distribution over observed property matrices conditioned on those features, $p(\mathbf{X}|\mathbf{F})$, where $p(\cdot)$ is a probability density function. These distributions can be dealt with separately: $p(\mathbf{F})$ specifies the number of features and the distribution over values associated with each feature, while $p(\mathbf{X}|\mathbf{F})$ determines how these features relate to the properties of objects. Our focus will be on $p(\mathbf{F})$, showing how such a prior can be defined without limiting the number of features.

We can break $\mathbf{F}$ into two components: a binary matrix $\mathbf{Z}$ indicating which features are possessed by each object, with $z_{ik} = 1$ if object $i$ has feature $k$ and $0$ otherwise, and a matrix $\mathbf{V}$ indicating the value of each feature for each object. $\mathbf{F}$ is the elementwise product of $\mathbf{Z}$ and $\mathbf{V}$, $\mathbf{F} = \mathbf{Z} \otimes \mathbf{V}$, as illustrated in Figure 1. In many latent feature models (e.g., PCA) objects have non-zero values on every feature, and every entry of $\mathbf{Z}$ is $1$. In *sparse* latent feature models (e.g., sparse PCA [14, 15]) only a subset of features take on non-zero values for each object, and $\mathbf{Z}$ picks out these subsets. A prior on $\mathbf{F}$ can be defined by specifying priors for $\mathbf{Z}$ and $\mathbf{V}$, with $p(\mathbf{F}) = P(\mathbf{Z})p(\mathbf{V})$, where $P(\cdot)$ is a probability mass function. We will focus on defining a prior on $\mathbf{Z}$, since the effective dimensionality of a latent feature model is determined by $\mathbf{Z}$. Assuming that $\mathbf{Z}$ is sparse, we can define a prior for infinite latent feature models by defining a distribution over infinite binary matrices. Our discussion of the Chinese restaurant process provides two desiderata for such a distribution: objects

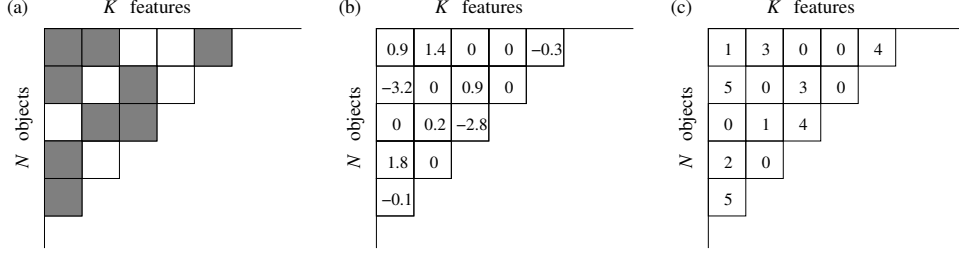

Figure 1: A binary matrix $\mathbf{Z}$, as shown in (a), indicates which features take non-zero values. Elementwise multiplication of $\mathbf{Z}$ by a matrix $\mathbf{V}$ of continuous values produces a representation like (b). If $\mathbf{V}$ contains discrete values, we obtain a representation like (c).

should be exchangeable, and posterior inference should be tractable. It also suggests a method by which these desiderata can be satisfied: start with a model that assumes a finite number of features, and consider the limit as the number of features approaches infinity.

## 2.1 A finite feature model

We have $N$ objects and $K$ features, and the possession of feature $k$ by object $i$ is indicated by a binary variable $z_{ik}$. The $z_{ik}$ form a binary $N \times K$ feature matrix, $\mathbf{Z}$. Assume that each object possesses feature $k$ with probability $\pi_k$, and that the features are generated independently. Under this model, the probability of $\mathbf{Z}$ given $\pi = \{\pi_1, \pi_2, \ldots, \pi_K\}$, is

$$P(\mathbf{Z}|\pi) = \prod_{k=1}^{K} \prod_{i=1}^{N} P(z_{ik}|\pi_k) = \prod_{k=1}^{K} \pi_k^{m_k}(1 - \pi_k)^{N-m_k}, \tag{1}$$

where $m_k = \sum_{i=1}^{N} z_{ik}$ is the number of objects possessing feature $k$. We can define a prior on $\pi$ by assuming that each $\pi_k$ follows a beta distribution, to give

$$\pi_k \mid \alpha \ \sim \text{Beta}(\tfrac{\alpha}{K}, 1)$$
$$z_{ik} \mid \pi_k \sim \text{Bernoulli}(\pi_k)$$

Each $z_{ik}$ is independent of all other assignments, conditioned on $\pi_k$, and the $\pi_k$ are generated independently. We can integrate out $\pi$ to obtain the probability of $\mathbf{Z}$, which is

$$P(\mathbf{Z}) \quad = \quad \prod_{k=1}^{K} \frac{\frac{\alpha}{K}\Gamma(m_k + \frac{\alpha}{K})\Gamma(N - m_k + 1)}{\Gamma(N + 1 + \frac{\alpha}{K})}. \tag{2}$$

This distribution is exchangeable, since $m_k$ is not affected by the ordering of the objects.

## 2.2 Equivalence classes

In order to find the limit of the distribution specified by Equation 2 as $K \to \infty$, we need to define equivalence classes of binary matrices – the analogue of partitions for class assignments. Our equivalence classes will be defined with respect to a function on binary matrices, $lof(\cdot)$. This function maps binary matrices to *left-ordered* binary matrices. $lof(\mathbf{Z})$ is obtained by ordering the columns of the binary matrix $\mathbf{Z}$ from left to right by the magnitude of the binary number expressed by that column, taking the first row as the most significant bit. The left-ordering of a binary matrix is shown in Figure 2. In the first row of the left-ordered matrix, the columns for which $z_{1k} = 1$ are grouped at the left. In the second row, the columns for which $z_{2k} = 1$ are grouped at the left of the sets for which $z_{1k} = 1$. This grouping structure persists throughout the matrix.

The *history* of feature $k$ at object $i$ is defined to be $(z_{1k}, \ldots, z_{(i-1)k})$. Where no object is specified, we will use *history* to refer to the full history of feature $k$, $(z_{1k}, \ldots, z_{Nk})$. We

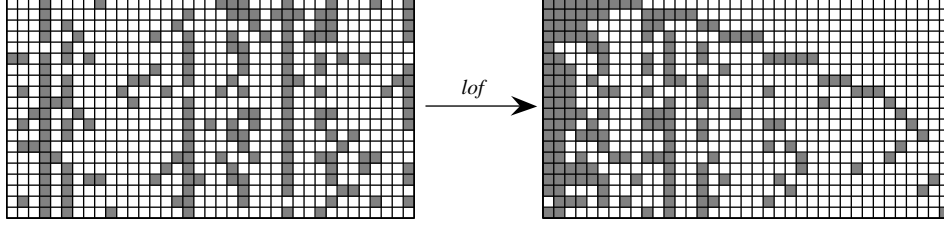

Figure 2: Left-ordered form. A binary matrix is transformed into a left-ordered binary matrix by the function $lof(\cdot)$. The entries in the left-ordered matrix were generated from the Indian buffet process with $\alpha = 10$. Empty columns are omitted from both matrices.

will individuate the histories of features using the decimal equivalent of the binary numbers corresponding to the column entries. For example, at object 3, features can have one of four histories: 0, corresponding to a feature with no previous assignments, 1, being a feature for which $z_{2k} = 1$ but $z_{1k} = 0$, 2, being a feature for which $z_{1k} = 1$ but $z_{2k} = 0$, and 3, being a feature possessed by both previous objects were assigned. $K_h$ will denote the number of features possessing the history $h$, with $K_0$ being the number of features for which $m_k = 0$ and $K_+ = \sum_{h=1}^{2^N-1} K_h$ being the number of features for which $m_k > 0$, so $K = K_0 + K_+$.

Two binary matrices $\mathbf{Y}$ and $\mathbf{Z}$ are $lof$-equivalent if $lof(\mathbf{Y}) = lof(\mathbf{Z})$. The $lof$-equivalence class of a binary matrix $\mathbf{Z}$, denoted $[\mathbf{Z}]$, is the set of binary matrices that are $lof$-equivalent to $\mathbf{Z}$. $lof$-equivalence classes play the role for binary matrices that partitions play for assignment vectors: they collapse together all binary matrices (assignment vectors) that differ only in column ordering (class labels). $lof$-equivalence classes are preserved through permutation of the rows or the columns of a matrix, provided the same permutations are applied to the other members of the equivalence class. Performing inference at the level of $lof$-equivalence classes is appropriate in models where feature order is not identifiable, with $p(\mathbf{X}|\mathbf{F})$ being unaffected by the order of the columns of $\mathbf{F}$. Any model in which the probability of $\mathbf{X}$ is specified in terms of a linear function of $\mathbf{F}$, such as PCA or CVQ, has this property. The cardinality of the $lof$-equivalence class $[\mathbf{Z}]$ is $\binom{K}{K_0 \ldots K_{2^N-1}} = \frac{K!}{\prod_{h=0}^{2^N-1} K_h!}$, where $K_h$ is the number of columns with full history $h$.

## 2.3 Taking the infinite limit

Under the distribution defined by Equation 2, the probability of a particular $lof$-equivalence class of binary matrices, $[\mathbf{Z}]$, is

$$P([\mathbf{Z}]) = \sum_{\mathbf{Z} \in [\mathbf{Z}]} P(\mathbf{Z}) = \frac{K!}{\prod_{h=0}^{2^N-1} K_h!} \prod_{k=1}^{K} \frac{\frac{\alpha}{K}\Gamma(m_k + \frac{\alpha}{K})\Gamma(N - m_k + 1)}{\Gamma(N + 1 + \frac{\alpha}{K})}. \quad (3)$$

Rearranging terms, and using the fact that $\Gamma(x) = (x-1)\Gamma(x-1)$ for $x > 1$, we can compute the limit of $P([\mathbf{Z}])$ as $K$ approaches infinity

$$\lim_{K \to \infty} \frac{\alpha^{K_+}}{\prod_{h=1}^{2^N-1} K_h!} \cdot \frac{K!}{K_0! K^{K_+}} \cdot \left( \frac{N!}{\prod_{j=1}^{N}(j + \frac{\alpha}{K})} \right)^K \cdot \prod_{k=1}^{K_+} \frac{(N - m_k)! \prod_{j=1}^{m_k-1}(j + \frac{\alpha}{K})}{N!}$$

$$= \frac{\alpha^{K_+}}{\prod_{h=1}^{2^N-1} K_h!} \cdot \quad 1 \quad \cdot \exp\{-\alpha H_N\} \quad \cdot \prod_{k=1}^{K_+} \frac{(N - m_k)!(m_k - 1)!}{N!}, \quad (4)$$

where $H_N$ is the $N$th harmonic number, $H_N = \sum_{j=1}^{N} \frac{1}{j}$. This distribution is infinitely exchangeable, since neither $K_h$ nor $m_k$ are affected by the ordering on objects. Technical details of this limit are provided in [13].

## 2.4 The Indian buffet process

The probability distribution defined in Equation 4 can be derived from a simple stochastic process. Due to the similarity to the Chinese restaurant process, we will also use a culinary metaphor, appropriately adjusted for geography. Indian restaurants in London offer buffets with an apparently infinite number of dishes. We will define a distribution over infinite binary matrices by specifying how customers (objects) choose dishes (features).

In our Indian buffet process (IBP), $N$ customers enter a restaurant one after another. Each customer encounters a buffet consisting of infinitely many dishes arranged in a line. The first customer starts at the left of the buffet and takes a serving from each dish, stopping after a Poisson($\alpha$) number of dishes. The $i$th customer moves along the buffet, sampling dishes in proportion to their popularity, taking dish $k$ with probability $\frac{m_k}{i}$, where $m_k$ is the number of previous customers who have sampled that dish. Having reached the end of all previous sampled dishes, the $i$th customer then tries a Poisson($\frac{\alpha}{i}$) number of new dishes. We can indicate which customers chose which dishes using a binary matrix $\mathbf{Z}$ with $N$ rows and infinitely many columns, where $z_{ik} = 1$ if the $i$th customer sampled the $k$th dish.

Using $K_1^{(i)}$ to indicate the number of new dishes sampled by the $i$th customer, the probability of any particular matrix being produced by the IBP is

$$P(\mathbf{Z}) = \frac{\alpha^{K_+}}{\prod_{i=1}^{N} K_1^{(i)}!} \exp\{-\alpha H_N\} \prod_{k=1}^{K_+} \frac{(N - m_k)!(m_k - 1)!}{N!}. \tag{5}$$

The matrices produced by this process are generally not in left-ordered form. These matrices are also not ordered arbitrarily, because the Poisson draws always result in choices of new dishes that are to the right of the previously sampled dishes. Customers are not exchangeable under this distribution, as the number of dishes counted as $K_1^{(i)}$ depends upon the order in which the customers make their choices. However, if we only pay attention to the $lof$-equivalence classes of the matrices generated by this process, we obtain the infinitely exchangeable distribution $P([\mathbf{Z}])$ given by Equation 4: $\frac{\prod_{i=1}^{N} K_1^{(i)}!}{\prod_{h=1}^{2^N-1} K_h!}$ matrices generated via this process map to the same left-ordered form, and $P([\mathbf{Z}])$ is obtained by multiplying $P(\mathbf{Z})$ from Equation 5 by this quantity. A similar but slightly more complicated process can be defined to produce left-ordered matrices directly [13].

## 2.5 Conditional distributions

To define a Gibbs sampler for models using the IBP, we need to know the conditional distribution on feature assignments, $P(z_{ik} = 1|\mathbf{Z}_{-(ik)})$. In the finite model, where $P(\mathbf{Z})$ is given by Equation 2, it is straightforward to compute this conditional distribution for any $z_{ik}$. Integrating over $\pi_k$ gives

$$P(z_{ik} = 1|\mathbf{z}_{-i,k}) = \frac{m_{-i,k} + \frac{\alpha}{K}}{N + \frac{\alpha}{K}}, \tag{6}$$

where $\mathbf{z}_{-i,k}$ is the set of assignments of other objects, not including $i$, for feature $k$, and $m_{-i,k}$ is the number of objects possessing feature $k$, not including $i$. We need only condition on $\mathbf{z}_{-i,k}$ rather than $\mathbf{Z}_{-(ik)}$ because the columns of the matrix are independent.

In the infinite case, we can derive the conditional distribution from the (exchangeable) IBP. Choosing an ordering on objects such that the $i$th object corresponds to the last customer to visit the buffet, we obtain

$$P(z_{ik} = 1|\mathbf{z}_{-i,k}) = \frac{m_{-i,k}}{N}, \tag{7}$$

for any $k$ such that $m_{-i,k} > 0$. The same result can be obtained by taking the limit of Equation 6 as $K \to \infty$. The number of new features associated with object $i$ should be

drawn from a Poisson($\frac{\alpha}{N}$) distribution. This can also be derived from Equation 6, using the same kind of limiting argument as that presented above.

## 3    A linear-Gaussian binary latent feature model

To illustrate how the IBP can be used as a prior in models for unsupervised learning, we derived and tested a linear-Gaussian latent feature model in which the features are binary. In this case the feature matrix $\mathbf{F}$ reduces to the binary matrix $\mathbf{Z}$. As above, we will start with a finite model and then consider the infinite limit.

In our finite model, the $D$-dimensional vector of properties of an object $i$, $\mathbf{x}_i$ is generated from a Gaussian distribution with mean $\mathbf{z}_i\mathbf{A}$ and covariance matrix $\mathbf{\Sigma}_X = \sigma_X^2 \mathbf{I}$, where $\mathbf{z}_i$ is a $K$-dimensional binary vector, and $\mathbf{A}$ is a $K \times D$ matrix of weights. In matrix notation, $E[\mathbf{X}] = \mathbf{ZA}$. If $\mathbf{Z}$ is a feature matrix, this is a form of binary factor analysis. The distribution of $\mathbf{X}$ given $\mathbf{Z}$, $\mathbf{A}$, and $\sigma_X$ is matrix Gaussian with mean $\mathbf{ZA}$ and covariance matrix $\sigma_X^2 I$, where $I$ is the identity matrix. The prior on $\mathbf{A}$ is also matrix Gaussian, with mean 0 and covariance matrix $\sigma_A^2 I$. Integrating out $\mathbf{A}$, we have

$$
p(\mathbf{X}|\mathbf{Z},\sigma_X,\sigma_A) = \frac{1}{(2\pi)^{ND/2}\sigma_X^{(N-K)D}\sigma_A^{KD}|\mathbf{Z}^T\mathbf{Z} + \frac{\sigma_X^2}{\sigma_A^2}\mathbf{I}|^{D/2}}
$$
$$
\exp\{-\frac{1}{2\sigma_X^2}\mathrm{tr}(\mathbf{X}^T(\mathbf{I} - \mathbf{Z}(\mathbf{Z}^T\mathbf{Z} + \frac{\sigma_X^2}{\sigma_A^2}\mathbf{I})^{-1}\mathbf{Z}^T)\mathbf{X})\}. \quad (8)
$$

This result is intuitive: the exponentiated term is the difference between the inner product of $\mathbf{X}$ and its projection onto the space spanned by $\mathbf{Z}$, regularized to an extent determined by the ratio of the variance of the noise in $\mathbf{X}$ to the variance of the prior on $\mathbf{A}$. It follows that $p(\mathbf{X}|\mathbf{Z},\sigma_X,\sigma_A)$ depends only on the non-zero columns of $\mathbf{Z}$, and thus remains well-defined when we take the limit as $K \to \infty$ (for more details see [13]).

We can define a Gibbs sampler for this model by computing the full conditional distribution

$$
P(z_{ik}|\mathbf{X},\mathbf{Z}_{-(i,k)},\sigma_X,\sigma_A) \propto p(\mathbf{X}|\mathbf{Z},\sigma_X,\sigma_A)P(z_{ik}|\mathbf{z}_{-i,k}). \quad (9)
$$

The two terms on the right hand side can be evaluated using Equations 8 and 7 respectively. The Gibbs sampler is then straightforward. Assignments for features for which $m_{-i,k} > 0$ are drawn from the distribution specified by Equation 9. The distribution over the number of new features for each object can be approximated by truncation, computing probabilities for a range of values of $K_1^{(i)}$ up to an upper bound. For each value, $p(\mathbf{X}|\mathbf{Z},\sigma_X,\sigma_A)$ can be computed from Equation 8, and the prior on the number of new features is Poisson($\frac{\alpha}{N}$).

We will demonstrate this Gibbs sampler for the infinite binary linear-Gaussian model on a dataset consisting of 100 $240 \times 320$ pixel images. We represented each image, $\mathbf{x}_i$, using a 100-dimensional vector corresponding to the weights of the mean image and the first 99 principal components. Each image contained up to four everyday objects – a \$20 bill, a Klein bottle, a prehistoric handaxe, and a cellular phone. Each object constituted a single latent feature responsible for the observed pixel values. The images were generated by sampling a feature vector, $\mathbf{z}_i$, from a distribution under which each feature was present with probability 0.5, and then taking a photograph containing the appropriate objects using a LogiTech digital webcam. Sample images are shown in Figure 3 (a).

The Gibbs sampler was initialized with $K_+ = 1$, choosing the feature assignments for the first column by setting $z_{i1} = 1$ with probability 0.5. $\sigma_A$, $\sigma_X$, and $\alpha$ were initially set to 0.5, 1.7, and 1 respectively, and then sampled by adding Metropolis steps to the MCMC algorithm. Figure 3 shows trace plots for the first 1000 iterations of MCMC for the number of features used by at least one object, $K_+$, and the model parameters $\sigma_A$, $\sigma_X$, and $\alpha$. All of these quantities stabilized after approximately 100 iterations, with the algorithm

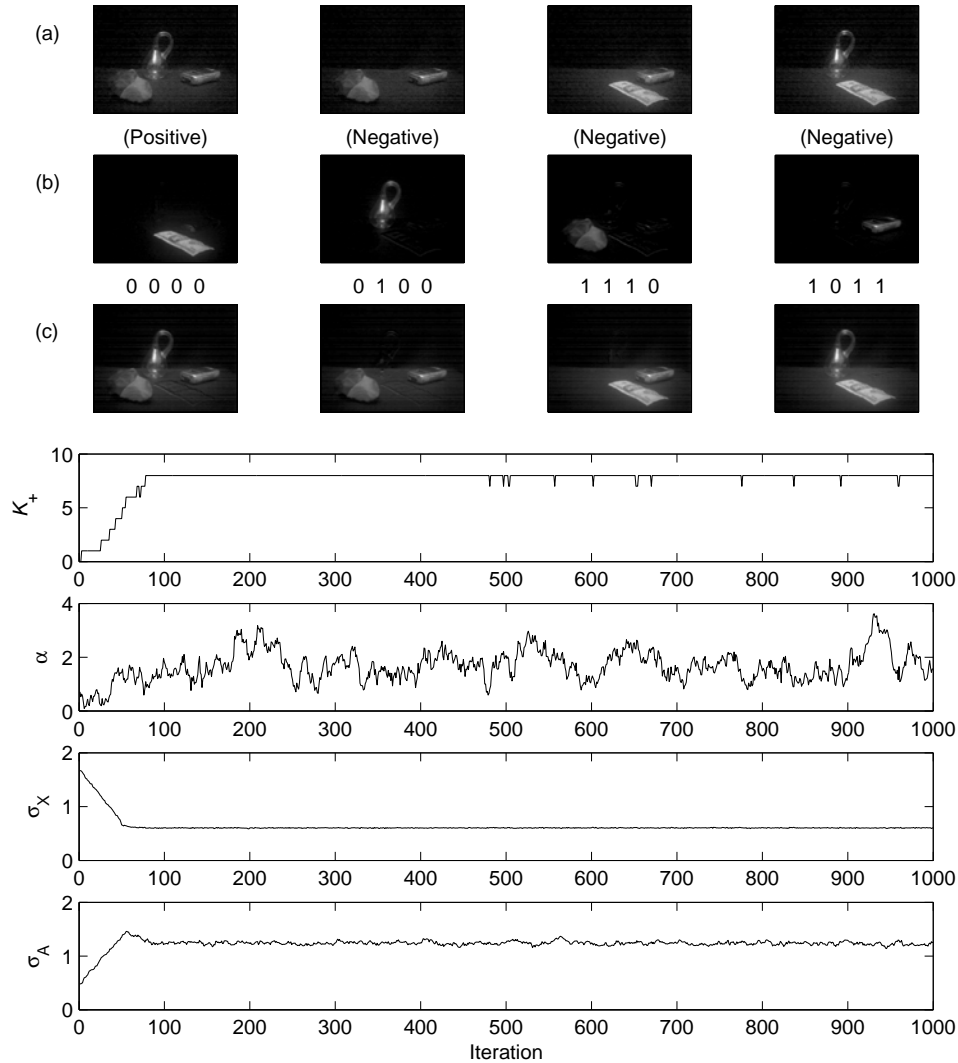

Figure 3: Data and results for the demonstration of the infinite linear-Gaussian binary latent feature model. (a) Four sample images from the 100 in the dataset. Each image had $320 \times 240$ pixels, and contained from zero to four everyday objects. (b) The posterior mean of the weights ($\mathbf{A}$) for the four most frequent binary features from the 1000th sample. Each image corresponds to a single feature. These features perfectly indicate the presence or absence of the four objects. The first feature indicates the presence of the $20 bill, the other three indicate the absence of the Klein bottle, the handaxe, and the cellphone. (c) Reconstructions of the images in (a) using the binary codes inferred for those images. These reconstructions are based upon the posterior mean of $\mathbf{A}$ for the 1000th sample. For example, the code for the first image indicates that the $20 bill is absent, while the other three objects are not. The lower panels show trace plots for the dimensionality of the representation ($K_+$) and the parameters $\alpha$, $\sigma_X$, and $\sigma_A$ over 1000 iterations of sampling. The values of all parameters stabilize after approximately 100 iterations.

finding solutions with approximately seven latent features. The four most common features perfectly indicated the presence and absence of the four objects (shown in Figure 3 (b)), and three less common features coded for slight differences in the locations of those objects.

## 4 Conclusion

We have shown that the methods that have been used to define infinite latent class models [6, 7, 8, 9, 10, 11, 12] can be extended to models in which objects are represented in terms of a set of latent features, deriving a distribution on infinite binary matrices that can be used as a prior for such models. While we derived this prior as the infinite limit of a simple distribution on finite binary matrices, we have shown that the same distribution can be specified in terms of a simple stochastic process – the Indian buffet process. This distribution satisfies our two desiderata for a prior for infinite latent feature models: objects are exchangeable, and inference remains tractable. Our success in transferring the strategy of taking the limit of a finite model from latent classes to latent features suggests that a similar approach could be applied with other representations, expanding the forms of latent structure that can be recovered through unsupervised learning.

## References

[1] N. Ueda and K. Saito. Parametric mixture models for multi-labeled text. In *Advances in Neural Information Processing Systems 15*, Cambridge, 2003. MIT Press.

[2] I. T. Jolliffe. *Principal component analysis*. Springer, New York, 1986.

[3] R. S. Zemel and G. E. Hinton. Developing population codes by minimizing description length. In *Advances in Neural Information Processing Systems 6*. Morgan Kaufmann, San Francisco, CA, 1994.

[4] Z. Ghahramani. Factorial learning and the EM algorithm. In *Advances in Neural Information Processing Systems 7*. Morgan Kaufmann, San Francisco, CA, 1995.

[5] C. E. Rasmussen and Z. Ghahramani. Occam's razor. In *Advances in Neural Information Processing Systems 13*. MIT Press, Cambridge, MA, 2001.

[6] C. Antoniak. Mixtures of Dirichlet processes with applications to Bayesian nonparametric problems. *The Annals of Statistics*, 2:1152–1174, 1974.

[7] M. D. Escobar and M. West. Bayesian density estimation and inference using mixtures. *Journal of the American Statistical Association*, 90:577–588, 1995.

[8] T. S. Ferguson. Bayesian density estimation by mixtures of normal distributions. In M. Rizvi, J. Rustagi, and D. Siegmund, editors, *Recent advances in statistics*, pages 287–302. Academic Press, New York, 1983.

[9] R. M. Neal. Markov chain sampling methods for Dirichlet process mixture models. *Journal of Computational and Graphical Statistics*, 9:249–265, 2000.

[10] C. Rasmussen. The infinite Gaussian mixture model. In *Advances in Neural Information Processing Systems 12*. MIT Press, Cambridge, MA, 2000.

[11] D. Aldous. Exchangeability and related topics. In *École d'été de probabilités de Saint-Flour, XIII—1983*, pages 1–198. Springer, Berlin, 1985.

[12] J. Pitman. Combinatorial stochastic processes, 2002. Notes for Saint Flour Summer School.

[13] T. L. Griffiths and Z. Ghahramani. Infinite latent feature models and the Indian buffet process. Technical Report 2005-001, Gatsby Computational Neuroscience Unit, 2005.

[14] A. d'Aspremont, L. El Ghaoui, I. Jordan, and G. R. G. Lanckriet. A direct formulation for sparse PCA using semidefinite programming. In *Advances in Neural Information Processing Systems 17*. MIT Press, Cambridge, MA, 2005.

[15] H. Zou, T. Hastie, and R. Tibshirani. Sparse principal component analysis. *Journal of Computational and Graphical Statistics*, in press.
